# Part-based Probabilistic Point Matching using Equivalence Constraints

**Graham McNeill, Sethu Vijayakumar**
Institute of Perception, Action and Behavior
School of Informatics, University of Edinburgh, Edinburgh, UK. EH9 3JZ
[graham.mcneill, sethu.vijayakumar]@ed.ac.uk

## Abstract

Correspondence algorithms typically struggle with shapes that display part-based variation. We present a probabilistic approach that matches shapes using independent part transformations, where the parts themselves are learnt during matching. Ideas from semi-supervised learning are used to bias the algorithm towards finding 'perceptually valid' part structures. Shapes are represented by unlabeled point sets of arbitrary size and a background component is used to handle occlusion, local dissimilarity and clutter. Thus, unlike many shape matching techniques, our approach can be applied to shapes extracted from real images. Model parameters are estimated using an EM algorithm that alternates between finding a soft correspondence and computing the optimal part transformations using Procrustes analysis.

## 1  Introduction

Shape-based object recognition is a key problem in machine vision and content-based image retrieval (CBIR). Over the last decade, numerous shape matching algorithms have been proposed that perform well on benchmark shape retrieval tests. However, many of these techniques share the same limitations: Firstly, they operate on contiguous shape boundaries (*i.e.* the ordering of the boundary points matters) and assume that every point on one boundary has a counterpart on the boundary it is being matched to (*c.f.* Fig. 1c). Secondly, they have no principled mechanism for handling occlusion, non-boundary points and clutter. Finally, they struggle to handle shapes that display significant *part-based variation*. The first two limitations mean that many algorithms are unsuitable for matching shapes extracted from real images; the latter is important since many common objects (natural and man made) display part-based variation.

Techniques that match unordered point sets (*e.g.* [1]) are appealing since they do not require ordered boundary information and can work with non-boundary points. The methods described in [2, 3, 4] can handle outliers, occlusions and clutter, but are not designed to handle shapes whose parts are independently transformed. In this paper, we introduce a probabilistic model that retains the desirable properties of these techniques but handles parts explicitly by learning the most likely part structure and correspondence simultaneously. In this framework, a *part* is defined as a set of points that undergo a common transformation. Learning these *variation-based parts* from scratch is an underconstrained problem. To address this, we incorporate prior knowledge about valid part assignments using two different mechanisms. Firstly, the distributions of our hierarchical mixture model are chosen so that the learnt parts are spatially localized. Secondly, ideas from semi-supervised learning [5] are used to encourage a perceptually meaningful part decomposition. The algorithm is introduced in Sec. 2 and described in detail in Sec. 3. Examples are given in Sec. 4 and a sequential approach for tackling model selection (the number of parts) and parameter initialization is introduced in Sec. 5.

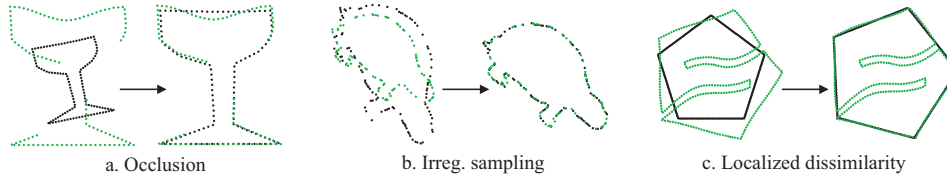

|  a. Occlusion | b. Irreg. sampling | c. Localized dissimilarity |

Figure 1: Examples of probabilistic point matching (PPM) using the technique described in [4]. In each case, the initial alignment and the final match are shown.

## 2  Part-based Point Matching (PBPM): Motivation and Overview

The PBPM algorithm combines three key ideas:

**Probabilistic point matching (PPM):** Probabilistic methods that find a *soft* correspondence between *unlabeled point sets* [2, 3, 4] are well suited to problems involving occlusion, absent features and clutter (Fig. 1).

**Natural Part Decomposition (NPD):** Most shapes have a *natural part decomposition* (NPD) (Fig. 2) and there are several algorithms available for finding NPDs (*e.g.* [6]). We note that in tasks such as object recognition and CBIR, the *query image* is frequently a template shape (*e.g.* a binary image or line drawing) or a high quality image with no occlusion or clutter. In such cases, one can apply an NPD algorithm prior to matching. Throughout this paper, it is assumed that we have obtained a sensible NPD *for the query shape only*[1] – it is not reasonable to assume that an NPD can be computed for each database shape/image.

**Variation-based Part Decomposition (VPD):** A different notion of parts has been used in computer vision [7], where a part is defined as a set of pixels that undergo the same transformations across images. We refer to this type of part decomposition (PD) as a *variation-based part decomposition* (VPD).

Given two shapes (*i.e.* point sets), PBPM matches them by applying a different transformation to each variation-based part of the *generating shape*. These variation-based parts are learnt during matching, where the known NPD of the *data shape* is used to bias the algorithm towards choosing a 'perceptually valid' VPD. This is achieved using the *equivalence constraint*

**Constraint 1 (C1):** *Points that belong to the same natural part should belong to the same variation-based part.*

As we shall see in Sec. 3, this influences the learnt VPD by changing the generative model from one that generates individual data points to one that generates natural parts (subsets of data points). To further increase the perceptual validity of the learnt VPD, we assume that variation-based parts are composed of spatially localized points of the generating shape.

PBPM aims to find the correct correspondence at the level of individual points, *i.e.* each point of the generating shape should be mapped to the correct position on the data shape despite the lack of an exact point wise correspondence (*e.g.* Fig. 1b). Soft correspondence techniques that achieve this using a *single nonlinear transformation* [2, 3] perform well on some challenging problems. However, the smoothness constraints used to control the nonlinearity of the transformation will prevent these techniques from selecting the *discontinuous transformations* associated with part-based movements. PBPM learns an independent linear transformation for each part and hence, can find the correct global match.

In relation to the point matching literature, PBPM is motivated by the success of the techniques described in [8, 2, 3, 4] on non-part-based problems. It is perhaps most similar to the work of Hancock and colleagues (*e.g.* [8]) in that we use 'structural information' about the point sets to constrain the matching problem. In addition to learning multiple parts and transformations, our work differs in the type of structural information used (the NPD rather then the Delauney triangulation) and the way in which this information is incorporated.

With respect to the shape-matching literature, PBPM can be seen as a novel correspondence technique for use with established NPD algorithms. Despite the large number of NPD algorithms, there

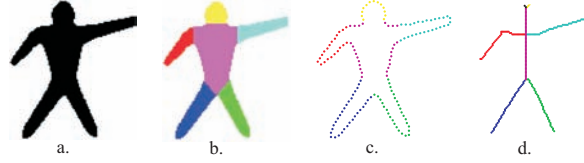

Figure 2: The natural part decomposition (NPD) (b-d) for different representations of a shape (a).

are relatively few NPD-based correspondence techniques. Siddiqi and Kimia show that the parts used in their NPD algorithm [6] correspond to specific types of shocks when shock graph representations are used. Consequently, shock graphs implicitly capture ideas about natural parts. The Inner-Distance method of Ling and Jacobs [9] handles part articulation without explicitly identifying the parts.

## 3 Part-based Point Matching (PBPM): Algorithm

### 3.1 Shape Representation

Shapes are represented by point sets of arbitrary size. The points need not belong to the shape boundary and the ordering of the points is irrelevant. Given a *generating shape* $\mathbf{X} = (\mathbf{x}_1, \mathbf{x}_2, \ldots, \mathbf{x}_M)^T \in \mathbb{R}^{M \times 2}$ and a *data shape* $\mathbf{Y} = (\mathbf{y}_1, \mathbf{y}_2, \ldots, \mathbf{y}_N)^T \in \mathbb{R}^{N \times 2}$ (generally $M \neq N$), our task is to compute the correspondence between $\mathbf{X}$ and $\mathbf{Y}$. We assume that an NPD of $\mathbf{Y}$ is available, expressed as a partition of $\mathbf{Y}$ into subsets (parts): $\mathbf{Y} = \bigcup_{l=1}^{L} \mathbf{Y}_l$.

### 3.2 The Probabilistic Model

We assume that a data point $\mathbf{y}$ is generated by the mixture model

$$p(\mathbf{y}) = \sum_{v=0}^{V} p(\mathbf{y}|v)\pi_v, \tag{1}$$

where $v$ indexes the variation-based parts. A uniform *background* component, $\mathbf{y}|(v{=}0) \sim \text{Uniform}$, ensures that all data points are explained to some extent and hence, robustifies the model against outliers. The distribution of $\mathbf{y}$ given a *foreground* component $v$ is itself a mixture model :

$$p(\mathbf{y}|v) = \sum_{m=1}^{M} p(\mathbf{y}|m, v)p(m|v), \quad v = 1, 2, \ldots, V, \tag{2}$$

with

$$\mathbf{y}|(m, v) \sim \mathcal{N}(T_v \mathbf{x}_m, \sigma^2 \mathbf{I}). \tag{3}$$

Here, $T_v$ is the transformation used to match points of part $v$ on $\mathbf{X}$ to points of part $v$ on $\mathbf{Y}$. Finally, we define $p(m|v)$ in such a way that the variation-based parts $v$ are forced to be *spatially coherent*:

$$p(m|v) = \frac{\exp\{-(\mathbf{x}_m - \boldsymbol{\mu}_v)^T \Sigma_v^{-1} (\mathbf{x}_m - \boldsymbol{\mu}_v)/2\}}{\sum_m \exp\{-(\mathbf{x}_m - \boldsymbol{\mu}_v)^T \Sigma_v^{-1} (\mathbf{x}_m - \boldsymbol{\mu}_v)/2\}}, \tag{4}$$

where $\boldsymbol{\mu}_v \in \mathbb{R}^2$ is a mean vector and $\Sigma_v \in \mathbb{R}^{2 \times 2}$ is a covariance matrix. In words, we identify $m \in \{1, \ldots, M\}$ with the point $\mathbf{x}_m$ that it indexes and assume that the $\mathbf{x}_m$ follow a bivariate Gaussian distribution. Since $m$ must take a value in $\{1, \ldots, M\}$, the distribution is normalized using the points $\mathbf{x}_1, \ldots, \mathbf{x}_M$ only. This assumption means that the $\mathbf{x}_m$ themselves are essentially generated by a GMM with $V$ components. However, this GMM is embedded in the larger model and maximizing the data likelihood will balance this GMM's desire for coherent parts against the need for the parts and transformations to explain the actual data (the $\mathbf{y}_n$). Having defined all the distributions, the next step is to estimate the parameters whilst making use of the known NPD of $\mathbf{Y}$.

### 3.3 Parameter Estimation

With respect to the model defined in the previous section, **C1** states that all $\mathbf{y}_n$ that belong to the same subset $\mathbf{Y}_l$ were generated by the same mixture component $v$. This requirement can be enforced using the technique introduced by Shental *et. al.* [5] for incorporating equivalence constraints

between data points in mixture models. The basic idea is to estimate the model parameters using the EM algorithm. However, when taking the expectation (of the complete log-likelihood) we now only sum over assignments of data points to components which are valid with respect to the constraints. Assuming that subsets and points within subsets are sampled i.i.d., it can be shown that the expectation is given by:

$$E = \sum_{v=0}^{V}\sum_{l=1}^{L} p(v|\mathbf{Y}_l)\log \pi_v + \sum_{v=0}^{V}\sum_{l=1}^{L}\sum_{\mathbf{y}_n \in \mathbf{Y}_l} p(v|\mathbf{Y}_l)\log p(\mathbf{y}_n|v). \tag{5}$$

Note that eq.(5) involves $p(v|\mathbf{Y}_l)$ – the responsibility of a component $v$ for a subset $\mathbf{Y}_l$, rather than the term $p(v|\mathbf{y}_n)$ that would be present in an unconstrained mixture model. Using the expression for $p(\mathbf{y}_n|v)$ in eq.(2) and rearranging slightly, we have

$$\begin{aligned} E &= \sum_{v=0}^{V}\sum_{l=1}^{L} p(v|\mathbf{Y}_l)\log \pi_v + \sum_{l=1}^{L} p(v{=}0|\mathbf{Y}_l)\log\{u^{|\mathbf{Y}_l|}\} \\ &+ \sum_{v=1}^{V}\sum_{l=1}^{L}\sum_{\mathbf{y}_n \in \mathbf{Y}_l} p(v|\mathbf{Y}_l)\log\left\{\sum_{m=1}^{M} p(\mathbf{y}_n|m,v)p(m|v)\right\}, \end{aligned} \tag{6}$$

where $u$ is the constant associated with the uniform distribution $p(\mathbf{y}_n|v{=}0)$. The parameters to be estimated are $\pi_v$ (eq.(1)), $\boldsymbol{\mu}_v$, $\Sigma_v$ (eq.(4)) and the transformations $T_v$ (eq.(3)). With the exception of $\pi_v$, these are found by maximizing the final term in eq.(6). For a fixed $v$, this term is the log-likelihood of data points $\mathbf{y}_1, \ldots, \mathbf{y}_N$ under a mixture model, with the modification that there is a weight, $p(v|\mathbf{Y}_l)$, associated with each data point. Thus, we can treat this subproblem as a standard maximum likelihood problem and derive the EM updates as usual. The resulting EM algorithm is given below.

**E-step.** Compute the *responsibilities* using the current parameters:

$$p(m|\mathbf{y}_n, v) = \frac{p(\mathbf{y}_n|m,v)p(m|v)}{\sum_m p(\mathbf{y}_n|m,v)p(m|v)}, \quad v = 1, 2, \ldots, V \tag{7}$$

$$p(v|\mathbf{Y}_l) = \frac{\pi_v \prod_{\mathbf{y}_n \in \mathbf{Y}_l} p(\mathbf{y}_n|v)}{\sum_v \pi_v \prod_{\mathbf{y}_n \in \mathbf{Y}_l} p(\mathbf{y}_n|v)} \tag{8}$$

**M-step.** Update the *parameters* using the responsibilities:

$$\pi_v = \frac{1}{L}\sum_{l=1}^{L} p(v|\mathbf{Y}_l) \tag{9}$$

$$\boldsymbol{\mu}_v = \frac{\sum_{n,m} p(v|\mathbf{Y}_{l,n})p(m|\mathbf{y}_n,v)\mathbf{x}_m}{\sum_{n,m} p(v|\mathbf{Y}_{l,n})p(m|\mathbf{y}_n,v)} \tag{10}$$

$$\Sigma_v = \frac{\sum_{n,m} p(v|\mathbf{Y}_{l,n})p(m|\mathbf{y}_n,v)(\mathbf{x}_m - \boldsymbol{\mu}_v)(\mathbf{x}_m - \boldsymbol{\mu}_v)^T}{\sum_{n,m} p(v|\mathbf{Y}_{l,n})p(m|\mathbf{y}_n,v)} \tag{11}$$

$$T_v = \arg\min_{T} \sum_{n,m} p(v|\mathbf{Y}_{l,n})p(m|\mathbf{y}_n,v)\|\mathbf{y}_n - T_v\mathbf{x}_m\|^2 \tag{12}$$

where $\mathbf{Y}_{l,n}$ is the subset $\mathbf{Y}_l$ containing $\mathbf{y}_n$. Here, we define $T_v\mathbf{x} \equiv s_v\Gamma_v\mathbf{x} + \mathbf{c}_v$, where $s_v$ is a scale parameter, $\mathbf{c}_v \in \mathbb{R}^2$ is a translation vector and $\Gamma_v$ is a 2D rotation matrix. Thus, eq.(12) becomes a *weighted Procrustes matching problem* between two points sets, each of size $N \times M$ – the extent to which $\mathbf{x}_m$ corresponds to $\mathbf{y}_n$ in the context of part $v$ is given by $p(v|\mathbf{Y}_{l,n})p(m|\mathbf{y}_n,v)$. This least squares problem for the optimal transformation parameters $s_v, \Gamma_v$ and $\mathbf{c}_v$ can be solved analytically [8]. The weights associated with the updates in eqs.(10-12) are similar to $p(v|\mathbf{y}_n)p(m|\mathbf{y}_n,v) = p(m,v|\mathbf{y}_n)$, the responsibility of the hidden variables $(m,v)$ for the observed data, $\mathbf{y}_n$. The difference is that $p(v|\mathbf{y}_n)$ is replaced by $p(v|\mathbf{Y}_{l,n})$, and hence, the impact of the equivalence constraints is propagated throughout the model.

The same fixed variance $\sigma^2$ (eq.(3)) is used in all experiments. For the examples in Sec. 4, we initialize $\pi_v$, $\boldsymbol{\mu}_v$ and $\Sigma_v$ by fitting a standard GMM to the $\mathbf{x}_m$. In Sec. 5, we describe a sequential algorithm that can be used to select the number of parts $V$ as well as provide initial estimates for all parameters.

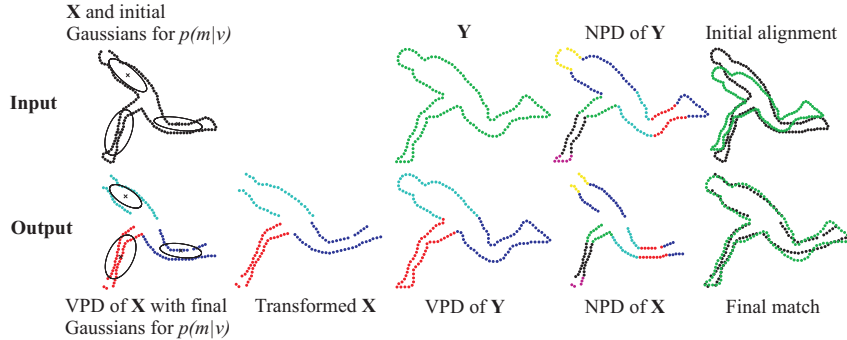

Figure 3: An example of applying PBPM with $V$=3.

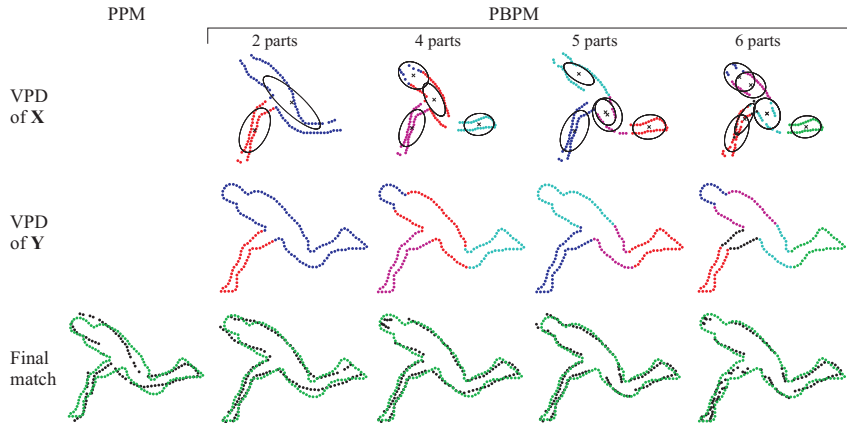

Figure 4: Results for the problem in Fig. 3 using PPM [4] and PBPM with $V = 2, 4, 5$ and $6$.

## 4   Examples

As discussed in Secs. 1 and 2, unsupervised matching of shapes with moving parts is a relatively unexplored area – particularly for shapes not composed of single closed boundaries. This makes it difficult to quantitatively assess the performance of our algorithm. Here, we provide illustrative examples which demonstrate the various properties of PBPM and then consider more challenging problems involving shapes extracted from real images. The number of parts, $V$, is fixed prior to matching in these examples; a technique for estimating $V$ is described in Sec. 5. To visualize the matches found by PBPM, each point $\mathbf{y}_n$ is assigned to a part $v$ using $\max_v p(v|\mathbf{y}_n)$. Points assigned to $v$=0 are removed from the figure. For each $\mathbf{y}_n$ assigned to some $v \in \{1, \ldots, V\}$, we find $m_n \equiv \arg\max_m p(m|\mathbf{y}_n, v)$ and assign $\mathbf{x}_{m_n}$ to $v$. Those $\mathbf{x}_m$ not assigned to any parts are removed from the figure. The means and the ellipses of constant probability density associated with the distributions $\mathcal{N}(\boldsymbol{\mu}_v, \Sigma_v)$ are plotted on the original shape $\mathbf{X}$. We also assign the $\mathbf{x}_m$ to natural parts using the known natural part label of the $\mathbf{y}_n$ that they are assigned to.

Fig. 3 shows an example of matching two human body shapes using PBPM with $V$=3. The learnt VPD is intuitive and the match is better than that found using PPM (Fig. 4). The results obtained using different values of $V$ are shown in Fig. 4. Predictably, the match improves as $V$ increases, but the improvement is negligible beyond $V$=4. When $V$=5, one of the parts is effectively repeated, suggesting that four parts is sufficient to cover all the interesting variation. However, when $V$=6 all parts are used and the VPD looks very similar to the NPD – only the lower leg and foot on each side are grouped together.

In Fig. 5, there are two genuine variation-based parts and $\mathbf{X}$ contains additional features. PBPM effectively ignores the extra points of $\mathbf{X}$ and finds the correct parts and matches. In Fig. 6, the left leg is correctly identified and rotated, whereas the right leg of $\mathbf{Y}$ is 'deleted'. We find that deletion from the generating shape tends to be very precise (*e.g.* Fig. 5), whereas PBPM is less inclined to delete points from the data shape when it involves breaking up natural parts (*e.g.* Fig. 6). This is

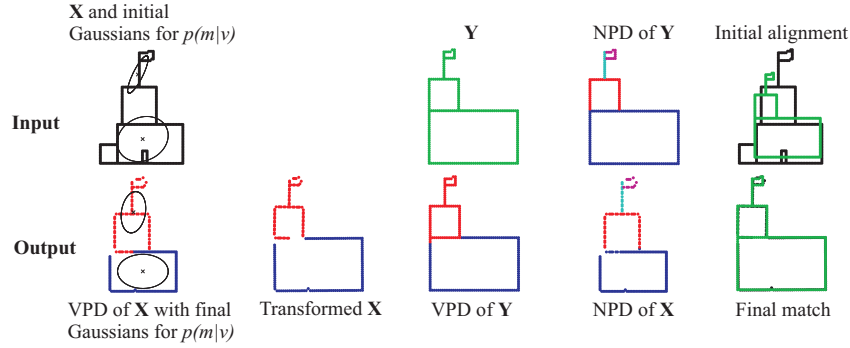

Figure 5: Some features of **X** are not present on **Y**; the main building of **X** is smaller and the tower is more central.

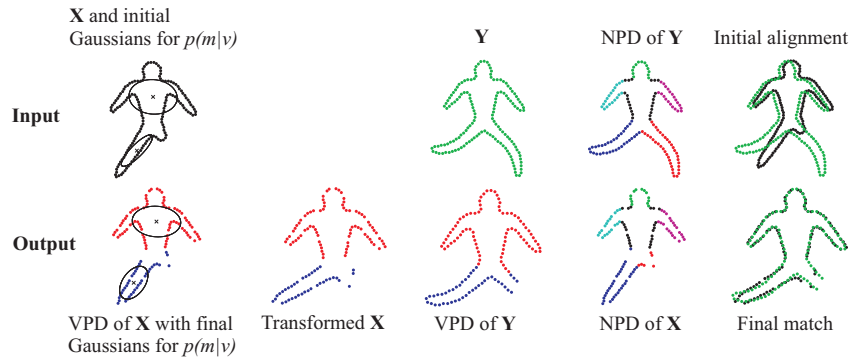

Figure 6: The left legs do not match and most of the right leg of **X** is missing.

largely due to the equivalence constraints trying to keep natural parts intact, though the value of the uniform density, $u$, and the way in which points are assigned to parts is also important.

In Figs. 7 and 8, a template shape is matched to the edge detector output from two real images. We have not focused on optimizing the parameters of the edge detector since the aim is to demonstrate the ability of PBPM to handle suboptimal shape representations. The correct correspondence and PDs is estimated in all cases, though the results are less precise for these difficult problems. Six parts are used in Fig. 8, but two of these are initially assigned to clutter and end up playing no role in the final match. The object of interest in **X** is well matched to the template using the other four parts. Note that the left shoulder is not assigned to the same variation-based part as the other points of the torso, *i.e.* the soft equivalence constraint has been broken in the interests of finding the best match.

We have not yet considered the choice of $V$. Figs. 4 (with $V{=}5$) and 8 indicate that it may be possible to start with more parts than are required and either allow extraneous parts to go unused or perhaps prune parts during matching. Alternatively, one could run PBPM for a range of $V$ and use a model selection technique based on a penalized log-likelihood function (*e.g.* BIC) to select a $V$. Finally, one could attempt to learn the parts in a sequential fashion. This is the approach considered in the next section.

## 5 Sequential Algorithm for Initialization

When part variation is present, one would expect PBPM with $V{=}1$ to find the most significant part and allow the background to explain the remaining parts. This suggests a sequential approach whereby a single part is learnt and removed from further consideration at each stage. Each new part/component should focus on data points that are currently explained by the background. This is achieved by modifying the technique described in [7] for fitting mixture models sequentially. Specifically, assume that the first part ($v{=}1$) has been learnt and now learn the second part using the

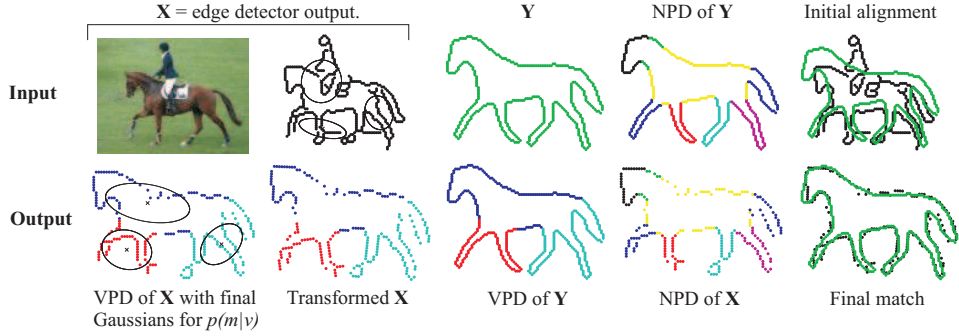

Figure 7: Matching a template shape to an object in a cluttered scene.

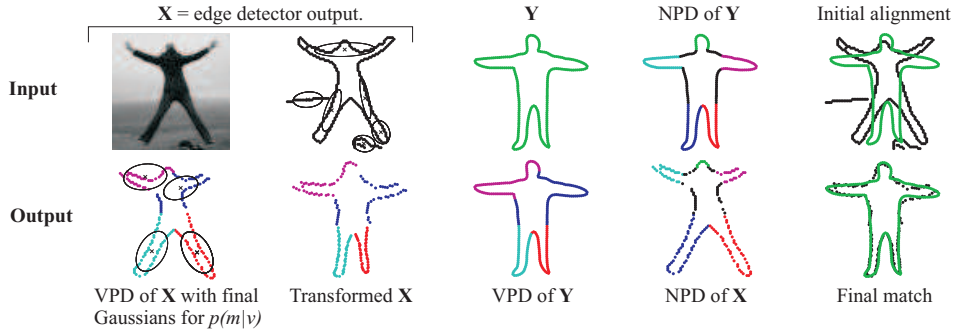

Figure 8: Matching a template shape to a real image.

weighted log-likelihood

$$\mathcal{J}_2 = \sum_{l=1}^{L} z_l^1 \log\{p(\mathbf{Y}_l|v{=}2)\pi_2 + u^{|\mathbf{Y}_l|}(1 - \pi_1 - \pi_2)\}. \tag{13}$$

Here, $\pi_1$ is known and

$$z_l^1 \equiv \frac{u^{|\mathbf{Y}_l|}(1 - \pi_1)}{p(\mathbf{Y}_l|v{=}1)\pi_1 + u^{|\mathbf{Y}_l|}(1 - \pi_1)} \tag{14}$$

is the responsibility of the background component for the subset $\mathbf{Y}_l$ after learning the first part – the superscript of $z$ indicates the number of components that have already been learnt. Using the modified log-likelihood in eq.(13) has the desired effect of forcing the new component ($v{=}2$) to explain the data currently explained by the uniform component. Note that we use the responsibilities for the subsets $\mathbf{Y}_l$ rather than the individual $\mathbf{y}_n$ [7], in line with the assumption that complete subsets belong to the same part. Also, note that eq.(13) is a weighted sum of log-likelihoods over the subsets, it cannot be written as a sum over data points since these are not sampled i.i.d. due to the equivalence constraints. Maximizing eq.(13) leads to similar EM updates to those given in eqs.(7)-(12). Having learnt the second part, additional components $v = 3, 4, \ldots$ are learnt in the same way except for minor adjustments to eqs.(13) and (14) to incorporate all previously learnt components. The sequential algorithm terminates when the uniform component is not significantly responsible for any data or the most recently learnt component is not significantly responsible for any data.

As discussed in [7], the sequential algorithm is expected to have fewer problems with local minima since the objective function will be smoother (a single component competes against a uniform component at each stage) and the search space smaller (fewer parameters are learnt at each stage). Preliminary experiments suggest that the sequential algorithm is capable of solving the model selection problem (choosing the number of parts) and providing good initial parameter values for the full model described in Sec. 3. Some examples are given in Figs. 9 and 10 – the initial transformations for each part are not shown. The outcome of the sequential algorithm is highly dependent on the value of the uniform density, $u$. We are currently investigating how the model can be made more robust to this value and also how the used $\mathbf{x}_m$ should be subtracted (in a probabilistic sense) at each step.

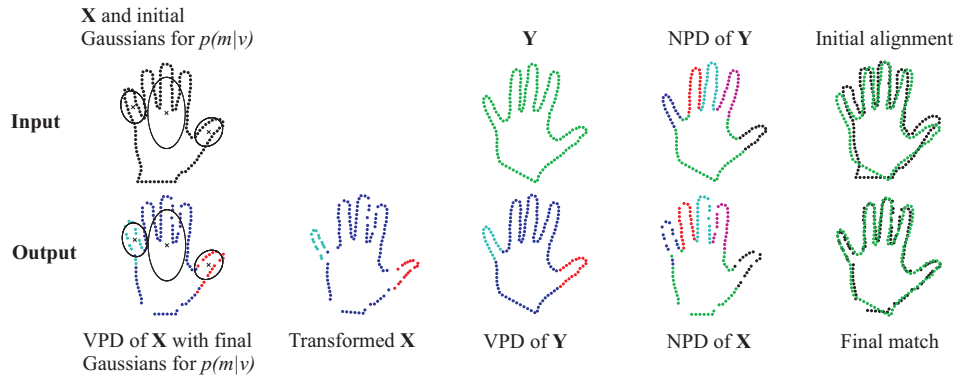

Figure 9: Results for PBPM; $V$ and initial parameters were found using the sequential approach.

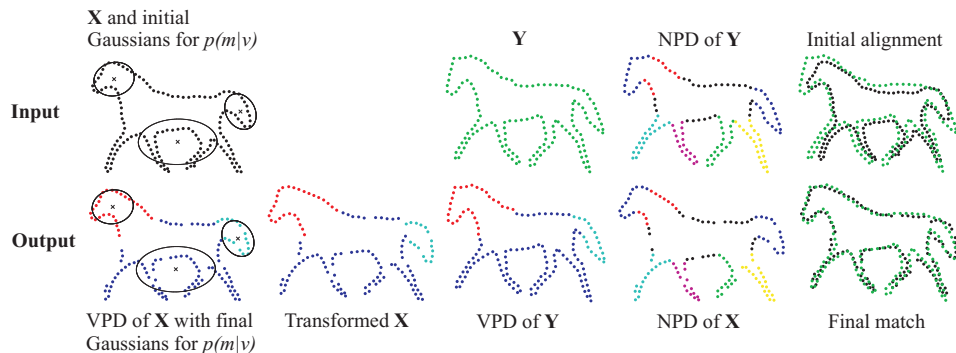

Figure 10: Results for PBPM; $V$ and initial parameters were found using the sequential approach.

## 6    Summary and Discussion

Despite the prevalence of part-based objects/shapes, there has been relatively little work on the associated correspondence problem. In the absence of class models and training data (*i.e.* the unsupervised case), this is a particularly difficult task. In this paper, we have presented a probabilistic correspondence algorithm that handles part-based variation by learning the parts and correspondence simultaneously. Ideas from semi-supervised learning are used to bias the algorithm towards finding a 'perceptually valid' part decomposition. Future work will focus on robustifying the sequential approach described in Sec. 5.

## Footnotes

[1]The NPDs used in the examples were constructed manually.

## References

[1] S. Belongie, J. Malik, and J. Puzicha. Shape matching and object recognition using shape contexts. *PAMI*, 24:509–522, 2002.

[2] H. Chui and A. Rangarajan. A new point matching algorithm for non-rigid registration. *Comp. Vis. and Image Understanding*, 89:114–141, 2003.

[3] Z. Tu and A.L. Yuille. Shape matching and recognition using generative models and informative features. In *ECCV*, 2004.

[4] G. McNeill and S. Vijayakumar. A probabilistic approach to robust shape matching. In *ICIP*, 2006.

[5] Noam Shental, Aharon Bar-Hillel, Tomer Hertz, and Daphna Weinshall. Computing Gaussian mixture models with EM using equivalence constraints. In *NIPS*. 2004.

[6] Kaleem Siddiqi and Benjamin B. Kimia. Parts of visual form: Computational aspects. *PAMI*, 17(3):239–251, 1995.

[7] M. Titsias. *Unsupervised Learning of Multiple Objects in Images*. PhD thesis, Univ. of Edinburgh, 2005.

[8] B. Luo and E.R. Hancock. A unified framework for alignment and correspondence. *Computer Vision and Image Understanding*, 92(26-55), 2003.

[9] H. Ling and D.W. Jacobs. Using the inner-distance for classification of ariculated shapes. In *CVPR*, 2005.
